# EMVP: Embracing Visual Foundation Model for Visual Place Recognition with Centroid-Free Probing

**Qibo Qiu**[1,2,∗]    **Shun Zhang**[3,∗]    **Haiming Gao**[3]
**Honghui Yang**[1]    **Haochao Ying**[4,†]    **Wenxiao Wang**[5]    **Xiaofei He**[1]
[1] State Key Lab of CAD&CG, Zhejiang University
[2] China Mobile (Zhejiang) Research & Innovation Institute, [3]Zhejiang Lab
[4]School of Public Health and Second Affiliated Hospital, Zhejiang University School of Medicine
[5]School of Software Technology, Zhejiang University
`qiuqibo_zju@zju.edu.cn, haochaoying@zju.edu.cn`

## Abstract

Visual Place Recognition (VPR) is essential for mobile robots as it enables them to retrieve images from a database closest to their current location. The progress of Visual Foundation Models (VFMs) has significantly advanced VPR by capturing representative descriptors in images. However, existing fine-tuning efforts for VFMs often overlook the crucial role of probing in effectively adapting these descriptors for improved image representation. In this paper, we propose the Centroid-Free Probing (CFP) stage, making novel use of second-order features for more effective use of descriptors from VFMs. Moreover, to control the preservation of task-specific information adaptively based on the context of the VPR, we introduce the Dynamic Power Normalization (DPN) module in both the recalibration and CFP stages, forming a novel Parameter Efficiency Fine-Tuning (PEFT) pipeline (EMVP) tailored for the VPR task. Extensive experiments demonstrate the superiority of the proposed CFP over existing probing methods. Moreover, the EMVP pipeline can further enhance fine-tuning performance in terms of accuracy and efficiency. Specifically, it achieves 93.9%, 96.5%, and 94.6% Recall@1 on the MSLS Validation, Pitts250k-test, and SPED datasets, respectively, while saving 64.3% of trainable parameters compared with the existing SOTA PEFT method. The code is available at `https://github.com/vincentqqb/EMVP`.

## 1   Introduction

Visual Place Recognition (VPR) is indispensable for mobile robots and autonomous vehicles, enabling key functions such as global localization [1], Simultaneous Localization and Mapping (SLAM) [2], and scene understanding [3]. VPR is often tackled as an image retrieval problem, where the objective is to match the query (an image representing the current location) with images from previously visited places. To achieve efficient matching, typically based on distance metrics like Euclidean distance, the VPR system aggregates the local descriptors of each image into a global descriptor. However, VPR faces unique challenges compared to conventional image retrieval tasks, including drastic changes in image perspectives, seasonal variations, and occlusions. Consequently, many researchers are dedicated to exploring robust local descriptors that exhibit invariance to these challenges. The advent of deep learning significantly brings VPR to a new stage characterized by enhanced robustness and improved accuracy [4; 5; 6; 7; 8; 9; 10; 11].

Despite the carefully designed pipelines, these methods typically involve training a model from scratch on environment-specific data. However, the diversity of application environments making

---

∗Equal contribution. † Corresponding author.

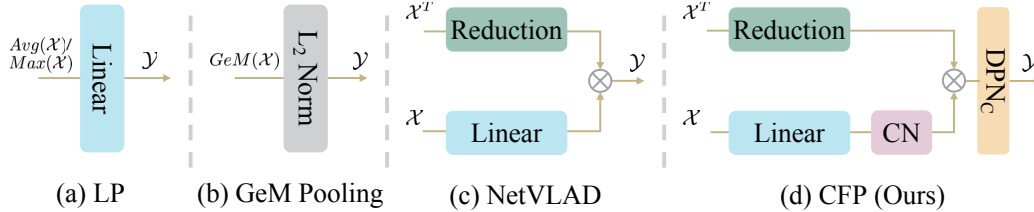

(a) LP          (b) GeM Pooling          (c) NetVLAD          (d) CFP (Ours)

Figure 1: Comparison of different probing methods. (a) The most popular Linear Probing (LP) in classification fine-tuning. (b) Generalized-Mean (GeM) pooling adapted by SelaVPR [13], which can be seen as a generalized form of first-order feature. (c) The NetVLAD operation simplified by SALAD [12]. (d) The proposed Centroid-Free Probing (CFP) which provides a theoretical and empirical justification for this simplification, fixing interpretability and performance issues that were present otherwise.

it challenging to collect sufficient data across different settings. To address this, a few recent studies [12; 13] have explored the potential of the Visual Foundation Model (VFM). These studies mainly focus on designing effective adapters within the backbone while only employing existing aggregation (probing) methods. *However, specific probing techniques for more effective fine-tuning in the VPR task remain largely untapped.* Typically, the most popular probing techniques mainly exploit first-order statistics of features, including Linear Probing (LP) for classification and Generalized-Mean (GeM) pooling for VPR, as shown in Figure 1 (a) and (b). However, second-order statistics have been well-proven important for fine-grained classification tasks [14], but LP and GeM are unable to capture them explicitly.

To this end, we revisit the classic method NetVLAD [4] as bilinear pooling (*i.e.*, the most basic second-order statistics), which aggregates local descriptors into a global one for fine-grained classification. However, NetVLAD requires a costly offline initialization of the semantic centroids, limiting its flexibility for fine-tuning on different datasets. In addition, inaccurate centroids can introduce inductive bias (for instance, initializing centroids in urban scenes but training or inferring in rural scenes), and affect the model's generalization ability. On the other hand, semantic centroids serve as priors for probing (aggregation), and simply removing them, as shown in Figure 1 (c), can lead to a decline in aggregation performance (detailed in Section 3.1). Delightfully, we observe that the explicit calculation of semantic centroids can be avoided when introducing a simple and effective **Constant Normalization (CN)** (detailed in Section 3.2). On this basis, a novel **Centroid-Free Probing (CFP)** stage is naturally introduced, which takes efforts of the second-order statistics of features.

Furthermore, due to the fact that VPR heavily relies on small overlapping regions between different images to make judgments when dealing with changes in image perspectives, preserving information from these discriminative regions is crucial. We design the **Dynamic Power Normalization (DPN)** module to adaptively control the preservation of task-specific information during the CFP stage, referred to as $DPN_C$. Moreover, due to the different training objectives, the general representation capability of pre-trained VFMs tends to focus more on foreground objects. However, the VPR task relies more on background regions such as the salient building. To address this, we insert DPN modules into the backbone, termed $DPN_R$, to enhance the preservation of information from these key background regions. With the backbone frozen, trainable $DPN_R$ modules adaptively control the preservation of task-specific background information in intermediate features, effectively contributing to Parameter Efficiency Fine-Tuning (PEFT). In the remainder of this paper, it is termed the recalibration stage. Thus, we propose a novel PEFT pipeline named EMVP by combining both recalibration and CFP stages. The main contributions of this paper can be summarized as follows:

- We are among the first to explore probing techniques for VPR tasks. By discussing the classical NetVLAD aggregation method, we demonstrate that avoiding costly and unstable initialization of semantic centroids allows for more effective fine-tuning of VFMs.

- We propose the novel DPN module to adaptively control the preservation of task-specific information in both recalibration and CFP stages. Based on the above, a more effective PEFT pipeline, named EMVP, is proposed.

- Extensive experiments demonstrate that the proposed EMVP pipeline significantly contributes to the more accurate VPR. Plenty of ablation studies have verified the effectiveness of indispensable components (*i.e.*, CFP, CN, $DPN_C$, and $DPN_R$).

## 2 Related Work

Visual Place Recognition (VPR) is typically studied as an image retrieval problem. Based on research topics, advanced VPR approaches can be categorized into improvements on the feature extraction of the backbone network [10], aggregation methods for local descriptors [4; 15; 16], the design of metric loss functions [17; 7], investigations into robustness against viewpoint changes [6; 18; 19], and others. This section discusses researches closely related to the proposed method.

### 2.1 Aggregation in Visual Place Recognition

Traditional VPR methods typically employ bag-of-visual-words [20; 21; 22], Vector of Locally Aggregated Descriptors (VLAD) [23; 24; 25] or Fisher Vectors (FV) [26; 27; 28] to aggregate local features such as SIFT [29] and SURF [30] into global ones. However, traditional methods for hand-crafted feature extraction are not data-driven. With the increase in data volume, these methods suffer from insufficient generalization and robustness. To address this problem, Arandjelovic *et al.* [4] proposed an end-to-end trainable generalized VLAD layer, NetVLAD, which greatly promotes the aggregation of features extracted from deep learning methods. Therefore, with the development of deep learning, NetVLAD becomes the most popular aggregation method for the VPR task [6; 8; 5]. Alternative techniques to NetVLAD include average/max pooling, R-MAC [31], and Generalized Mean (GeM) [15; 32]. Although these methods exhibit promising effectiveness in retrieving images, they regularly demonstrate inferior performance compared to NetVLAD in the VPR task [16; 33]. Benefiting from the emergence of the Visual Foundation Model (VFM) and embodied AI, mobile robots have progressed greatly in visual tasks. For instance, the latest research [34] shows that the combination of a self-supervised pre-trained ViT model (*i.e.*, DINOv2) and the unsupervised aggregation method VLAD exhibits robust zero-shot VPR performance. This motivates us to explore the cooperation between supervised NetVLAD and VFM for better accuracy.

In this paper, we avoid the costly explicit calculation of semantic centroids required by NetVLAD, by introducing a simple and effective Constant Normalization (CN) (detailed in Section 3.2). On this basis, a novel Centroid-Free Probing (CFP) stage including the $DPN_C$ module is proposed to employ second-order features when fine-tuning a VFM for better VPR performance.

**Differences between CFP and SALAD [12].** SALAD also avoids explicit calculation of the semantic centroids when full fine-tuning a VFM, directly aggregating local descriptors with a summation. More impressively, we show that CFP admits a theoretical and empirical justification for this simplification, fixing interpretability and performance issues that were present otherwise. Furthermore, a novel PEFT pipeline (*i.e.*, EMVP) tailored for the VPR task is proposed, innovatively employing the same DPN module in both recalibration and CFP stages for task-specific information preservation. Therefore, superior performance can be achieved with minimized trainable parameters.

### 2.2 Fine-tuning of Visual Foundation Model

Inspired by the remarkable language generation capabilities and interactivity demonstrated by the GPT series [35], pre-trained on expansive text corpora, subsequent researches in VFMs has flourished. VFMs, including SAM [36], DINOv2 [37], and the multi-modal CLIP [38], exhibit notable visual generalizability and robustness in the realm of 2D image recognition. Accordingly, there are also plenty of researches that focus on fine-tuning these VFMs in diverse downstream tasks, including adapters [39], prompt tuning [40], Low-Rank Adaptation (LoRA) [41], *etc*. Nonetheless, the mainstream efforts of PEFT primarily aim to improve parameter efficiency and mitigate overfitting, neglecting to learn a task-dependent classification head. As a result, efforts of tuning a linear classifier rely solely on the first-order features, *i.e.*, LP, suffering from inferior performance. To address this, Gao *et al.* [42] propose a novel Moment Probing (MP) method, which firstly leverages the second-order features that are rich in statistical information for PEFT. The second-order features used in MP are the covariance representation of the first-order features extracted by a single branch. In contrast, the second-order covariance matrix used in the proposed CFP stage comes from first-order

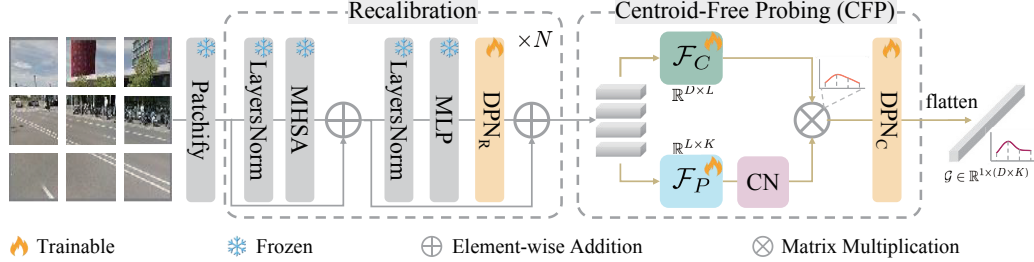

Figure 2: Overall pipeline of the proposed EMVP, including recalibration and CFP stages. Feature matrices from the two branches (*i.e.*, $\mathcal{F}_C$ and $\mathcal{F}_P$) are multiplied to obtain fine-grained features for the improved VPR performance. The Dynamic Power Normalization (DPN) layer can be inserted into both the recalibration and CFP stages to enhance the task-specific fine-tuning performance.

features extracted by two different branches. Note that both CFP and MP make use of second-order features, but they have different theoretical bases and excel in different tasks.

## 3 Method

The overall EMVP pipeline is illustrated in Figure 2, which can be further divided into the recalibration and Centroid-Free Probing (CFP) stages. The idea of CFP originates from NetVLAD and facilitates a more effective adaptation of VFMs to VPR tasks, with details elaborated in Section 3.1 and 3.2. Additionally, we propose the Dynamic Power Normalization (DPN) module in Section 3.3, and incorporate it into both the recalibration and CFP stages to enhance fine-tuning performance, thereby making the features extracted by the backbone network more task-specific.

### 3.1 Preliminaries

Global aggregation is crucial for robust and accurate visual place recognition, which aggregates local descriptors into a fixed-size global one. NetVLAD [4] is one of the most popular global aggregations based on the bag-of-visual-words [20] theory. Specifically, NetVLAD produces a statistic embedding for each semantic centroid, which represents the sum of distances from the semantic centroid to the assigned local descriptors as follows:

$$V_k = \sum_{i=1}^{L} p_{ik}(X_i - C_k), \qquad (1)$$

where $C_k$ and $V_k$ represent the embedding and the statistic embedding of the $k$-th semantic centroid, respectively. $L$ denotes the number of local descriptors in an image, and $X_i$ represents the $i$-th local descriptor. $K$ is the number of total semantic centroids. $p_{ik}$ indicates the probability that $X_i$ is assigned to $C_k$, which can be predicted by linear layers. The calculation of global descriptor can be summarized as:

$$\hat{\mathcal{G}} = NetVLAD(\mathcal{X}, \mathcal{C}) = cat(\mathcal{V}, dim = 0), \qquad (2)$$

where $\mathcal{X} = \{X_1, X_2, \ldots, X_L\} \in \mathbb{R}^{L \times M}$ represents all local descriptors from an image. $\hat{\mathcal{G}}$ is the global descriptor for visual place recognition, which is obtained by concatenating $\mathcal{V} = \{V_1, V_2, \ldots, V_K\} \in \mathbb{R}^{K \times M}$.

Note that the semantic centroids $\mathcal{C} = \{C_1, C_2, \ldots, C_K\} \in \mathbb{R}^{K \times M}$ are also trainable, and the initialization of $\mathcal{C}$ depends on an offline clustering process: Initially, an off-the-shelf backbone model is used to extract local descriptors from each image, involving a costly iteration through a pre-collected image set. Subsequently, $k$-means clustering is applied to these descriptors to obtain the semantic centroids $\mathcal{C}$. However, the offline clustering process overlooks an issue: both the pre-collected image set and off-the-shelf backbone model should be compatible with the training set and trained VPR model. In other words, the fine-tuning performance is sensitive to the initialization process [34]. Therefore, it motivates us to explore methods to avoid the explicitly computation of the semantic centroids and further stimulate the potential of VFM in the field of VPR.

## 3.2 Centroid-Free Probing

As discussed by [43], the NetVLAD operation can be written as a bilinear pooling model. Specifically, different from the typical centroid-wise calculation pipeline (*i.e.*, Equation 1 and 2), it can be calculated by accumulating local-wise descriptors as follows:

$$\mathcal{G} = NetVLAD(\mathcal{X}, \mathcal{C})$$

$$= \sum_{i=1}^{L} \left( [X_i - C_1; X_i - C_2; ...; X_i - C_K] \odot [\underbrace{p_{i1}, p_{i1}, ... p_{i1}}_{D}; ...; p_{iK}, p_{iK}, ... p_{iK}] \right), \quad (3)$$

where $P_i = [p_{i1}, p_{i2}, ..., p_{iK}]$ and $\odot$ indicates element-wise product. Note that Equation 3 contains the semantic centroids $\mathcal{C}$, which indicates that the corresponding costly and unstable offline initialization discussed in Section 3.1 is still needed.

We observed that since semantic centroids $\mathcal{C}$ are shared when extracting global descriptors for different images, if the value of $\sum_{i=1}^{L} P_i$ can be guaranteed to be constant, then the term after the minus sign in Equation 4 can be treated as a negligible constant term. Thus, the explicit calculation of $\mathcal{C}$ can be avoided during the training process. In this paper, we resort to post normalization methods (*e.g.*, softmax and $\ell_2$ normalization) to constrain the value of $\sum_{i=1}^{L} P_i$ to be constant, which is referred as Constant Normalization (CN) hereafter. The above simplification is formulated as follows:

$$\mathcal{G} = \sum_{i=1}^{L} X_i^T \times P_i - \underbrace{[C_1; C_2; ...; C_K] \odot \left( \sum_{i=1}^{L} P_i \right).expand(K, D)}_{constant}. \quad (4)$$

Therefore, we can represent the global descriptor as follows, by omitting the constant term:

$$\mathcal{G} = \sum_{i=1}^{L} X_i^T \times P_i = \mathcal{X}^T \mathcal{P} \approx \mathcal{F}_C(\mathcal{X})^T \mathcal{F}_P(\mathcal{X}), \quad (5)$$

where the probability matrix $\mathcal{P} = \{P_1, P_2, \ldots, P_L\} \in \mathbb{R}^{L \times K}$ represents the probability from $L$ local descriptors to $K$ semantic centroids. Note that, local descriptors $\mathcal{X}$ are fed to linear layers $\mathcal{F}_C$ and $\mathcal{F}_P$, which are designed for dimensionality reduction and computation of probability matrix $\mathcal{P}$, respectively. In the context of fine-tuning VFMs, we refer to this simplified aggregation operation as Centroid-Free Probing (CFP), as illustrated in Figure 2. It implicitly leverages the priors brought by centroids through a bilinear design, while avoiding explicit centroid initialization and training.

## 3.3 Dynamic Power Normalization in CFP and Recalibration

As VPR heavily relies on small overlapping regions to handle perspective changes, preserving information from these discriminative regions is essential. In this paper, we resort to post-processing methods to preserve task-specific information from these regions, thereby enhancing the robustness of second-order features. Extensive advanced studies [44; 14; 45] have verified that using the Matrix Power Normalization (MPN) method for feature post-processing after bilinear pooling can significantly improve downstream task performance. Typical MPN can be described as follows:

$$\mathcal{Y} = \left\| sign(\mathcal{G}) |\mathcal{G}|^{\alpha} \right\|_2, \quad (6)$$

where $sign(\mathcal{G})$ represents the sign of $\mathcal{G}$, $|\mathcal{G}|$ indicates the absolute value of $\mathcal{G}$, $\alpha$ is a scalar, and $\|\cdot\|_2$ denotes the $\ell_2$ normalization operation. MPN can effectively control the preservation of task-specific information during the training process by adjusting the value of $\alpha$. In particular, as $\alpha \longmapsto 0$, the normalized representation $\mathcal{Y}$ tends toward becoming an all-ones matrix. As $\alpha \longmapsto 1$, information in $\mathcal{G}$ will be gradually preserved [45]. In the classic MPN method, the value of $\alpha$ is predefined, and all images share the same value of $\alpha$. It can even be deduced from the perspective of Maximum Likelihood Estimation (MLE) that the theoretically optimal value for $\alpha$ is 0.5 [14].

Considering potential significant distribution differences between fine-tuning and inference datasets, in this paper, we lean towards $\alpha$ being learnable based on the context adaptively. Specifically, we design the Dynamic Power Normalization (DPN) module in the CFP stage to compute the value of

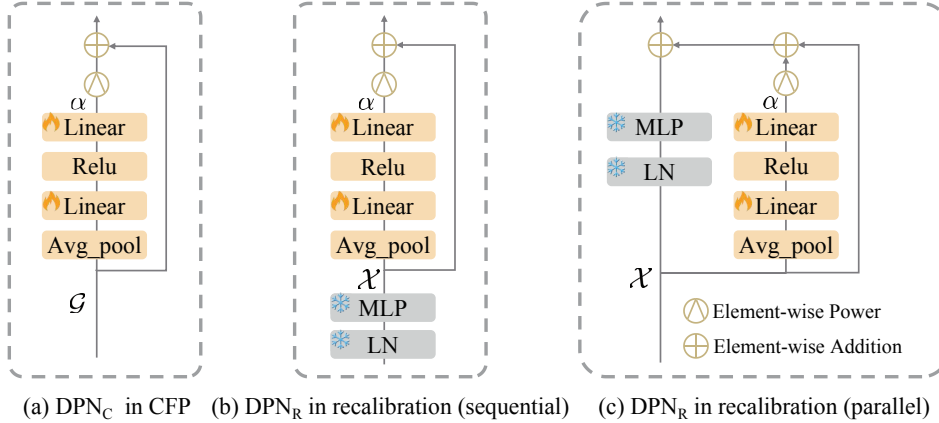

(a) DPN$_\mathrm{C}$ in CFP    (b) DPN$_\mathrm{R}$ in recalibration (sequential)    (c) DPN$_\mathrm{R}$ in recalibration (parallel)

Figure 3: The DPN module can be placed in both CFP and recalibration stages, which is indicated by DPN$_\mathrm{C}$ and DPN$_\mathrm{R}$, respectively. More importantly,, it can be inserted into the Transformer blocks sequentially and parallelly.

$\alpha$ based on the value of $\mathcal{G}$ as shown in Figure 3(a). Thus, each image can have a different level of task-specific information preservation, enabling more flexible fine-tuning.

With the same consideration, we attempt to use DPN to adaptively preserve task-specific information in recalibration for Parameter Efficiency Fine-Tuning (PEFT), while keeping the backbone network frozen. Given the unique nature of the VPR task: the background region in images (*e.g.*, the building), which may be irrelevant in other tasks, serves as a crucial clue for place recognition. However, due to the difference in training objectives, a pre-trained VFM may overlook these background regions. Therefore, we employ the DPN to enhance the representation of distinctive background regions, and effectively leveraging the representation capabilities of a VFM while minimizing recalibration, as depicted in Figure 3 (b) and 3 (c).

## 4 Experiments

### 4.1 Experimental Settings

**Choice of Visual Foundation Model.** To make effective use of large-scale unlabeled data, self-supervised transformers are becoming increasingly popular for training a VFM. Two self-supervised learning paradigms have demonstrated superior performance for ViT pre-training: contrastive learning-based (*e.g.*, DINO [46]), and masked image modeling-based (*e.g.*, MAE [47]). There are also multi-modal foundation models for robust feature extraction (*e.g.*, CLIP [38]). Recent advanced research [48] demonstrated that ViTs pre-trained by the contrastive learning paradigm can produce more universal path-wise representations. In this paper, the ViT model pre-trained by DINOv2, a follow-up of DINO, is adopted as the VFM, which has been verified to have good performance in fine-grained tasks (*e.g.*, depth estimation [37] and VPR [34]). Specifically, three VPR models are fine-tuned based on ViT-S, ViT-B, and ViT-L, named EMVP-S, EMVP-B, and EMVP-L, respectively.

**Implementation Details.** Both the $\mathcal{F}_C$ and $\mathcal{F}_P$ branches are implemented by a two-layer MLP network. For a fair comparison with SALAD, the output dimensions of $\mathcal{F}_C$ and $\mathcal{F}_P$ are 128 and 64, respectively. We employ the softmax operation to normalize the output features of $\mathcal{F}_P$. We implement two versions of the DPN$_\mathrm{R}$ module (*i.e.*, sequential and parallel DPN$_\mathrm{R}$), as shown in Figure 3, for the recalibration of intermediate features. In the parallel version, the original feature is preserved through an independent branch, and updated context is aggregated via element-wise addition. In contrast, the sequential version is equivalent to adding a few extra layers to the backbone. For more implementation details, please refer to Appendix A.1.

**Datasets.** Many advanced efforts [33; 16; 12] have shown that models trained on the GSV-Cities [33] dataset exhibit strong generalization across various VPR datasets, such as MSLS Validation [49], Pittsburgh30k-test [50], Pittsburgh250k-test [50], Nordland [51; 52], and SPED [53]. Following the training and validation protocol of these studies, we fine-tune the VPR model on the GSV-Cities

Table 1: Comparison with state-of-the-art methods. $^{\flat}$ denotes models trained on the GSV-Cities dataset. Due to the high quality of annotations in GSV-Cities, results from models marked with $^{\flat}$ generally outperform those from their corresponding papers. In contrast, results from models without $^{\flat}$ are reported in their respective papers.

(a) Comparison with single-stage methods.

| Method | MSLS Val | | | NordLand⋆ [52] | | | Pitts250k-test | | | SPED | | |
|---|---|---|---|---|---|---|---|---|---|---|---|---|
| | R@1 | R@5 | R@10 | R@1 | R@5 | R@10 | R@1 | R@5 | R@10 | R@1 | R@5 | R@10 |
| SPE-VLAD$^{\flat}$ [54] | 78.2 | 86.8 | 88.8 | 25.5 | 40.1 | 46.1 | 89.9 | 96.1 | 97.3 | 73.1 | 85.5 | 88.7 |
| Gated NetVLAD$^{\flat}$ [55] | 82.0 | 88.9 | 91.4 | 34.4 | 50.4 | 57.7 | 89.7 | 95.9 | 97.1 | 75.6 | 87.1 | 90.8 |
| NetVLAD$^{\flat}$ [4] | 82.6 | 89.6 | 92.0 | 32.6 | 47.1 | 53.3 | 90.5 | 96.2 | 97.4 | 78.7 | 88.3 | 91.4 |
| Conv-AP$^{\flat}$ [33] | 83.4 | 90.5 | 92.3 | 38.2 | 54.8 | 61.2 | 92.4 | 97.4 | 98.4 | 80.1 | 90.3 | 93.6 |
| CosPlace$^{\flat}$ [17] | 83.0 | 89.9 | 91.8 | 34.4 | 49.9 | 56.5 | 91.5 | 96.9 | 97.9 | 75.3 | 85.9 | 88.6 |
| MixVPR$^{\flat}$ [16] | 88.0 | 92.7 | 94.6 | 58.4 | 74.6 | 80.0 | 94.6 | 98.3 | 99.0 | 85.2 | 92.1 | 94.6 |
| EigenPlaces [19] | 89.3 | 93.7 | 95.0 | 54.4 | 68.8 | 74.1 | 94.1 | 98.0 | 98.7 | 69.9 | 82.9 | 87.6 |
| SALAD$^{\flat}$ [12] | 92.2 | 96.4 | 97.0 | 76.0 | 89.2 | 92.0 | 95.1 | 98.5 | 99.1 | 92.1 | 96.2 | 96.5 |
| **EMVP-L$^{\flat}$ (Ours)** | **93.9** | **97.3** | **97.6** | **78.4** | **89.7** | **92.4** | **96.5** | **99.1** | **99.5** | **94.6** | **97.5** | **98.4** |

(b) Comparison with two-stage methods, which include a re-ranking stage indicated by $^{\sharp}$.

| Method | MSLS Val | | | NordLand⋆⋆ [51] | | | Pitts30k-test | | |
|---|---|---|---|---|---|---|---|---|---|
| | R@1 | R@5 | R@10 | R@1 | R@5 | R@10 | R@1 | R@5 | R@10 |
| SP-SuperGlue$^{\sharp}$ [56] | 78.1 | 81.9 | 84.3 | 25.8 | 35.4 | 38.2 | 87.2 | 94.8 | 96.4 |
| Patch NetVLAD$^{\sharp}$ [5] | 79.5 | 86.2 | 87.7 | 51.6 | 60.1 | 62.8 | 88.7 | 94.5 | 95.9 |
| DELG$^{\sharp}$ [57] | 83.2 | 90.0 | 91.1 | 51.3 | 66.8 | 69.8 | 89.9 | 95.4 | 96.7 |
| TransVPR$^{\sharp}$ [10] | 86.8 | 91.2 | 92.4 | 58.8 | 75.0 | 78.7 | 89.0 | 94.9 | 96.2 |
| R2Former$^{\sharp}$ [11] | 89.7 | 95.0 | 96.2 | 77.0 | 89.0 | 91.9 | 91.1 | 95.2 | 96.3 |
| SelaVPR$^{\sharp}$ [13] | 90.8 | 96.4 | 97.2 | 85.2 | 95.5 | 98.5 | 92.8 | 96.8 | 97.7 |
| TransVPR w/o re-ranking [10] | 70.8 | 85.1 | 89.6 | 15.9 | 38.6 | 49.4 | 73.8 | 88.1 | 91.9 |
| SelaVPR (gobal) [13] | 87.7 | 95.8 | 96.6 | 72.3 | 89.4 | 94.4 | 90.2 | 96.1 | 97.1 |
| **EMVP-L$^{\flat}$ (Ours)** | **93.9** | **97.3** | **97.6** | **88.7** | **97.3** | **99.3** | **94.0** | **97.5** | **98.2** |

dataset, which is pre-trained by the DINOv2 pipeline. Subsequently, Recall@K (*i.e.*, R@1, R@5, and R@10) is evaluated across various VPR datasets as the performance metric.

**Model Selection.** We involve 30 epochs for fine-tuning models, and select the one with the highest R@1 on the Pittsburgh30k-val dataset for further evaluation on other test datasets. To fairly compare model performance, we repeat the aforementioned process 5 times and calculate the average metrics as the final test results.

## 4.2 Main Results

We report the performance comparison with SOTA in Table 1 and analyze the results as follows. First, in typical VPR methods, a re-ranking stage is typically incorporated for retrieved images to enhance the final performance as post-processing. This is primarily due to the inherent noise in the training dataset, which is subject to changes in image perspectives, seasonal variations, occlusion, and other factors. We illustrate the comparison through the TransVPR algorithm with or without the re-ranking stage as an example. Second, supported by the high-quality GSV-Cities dataset, methods (*e.g.*, MixVPR) without re-ranking achieve comparable performance to those with re-ranking stages. Third, with the further support of DINOv2, the algorithms without re-ranking achieve new levels of accuracy and robustness, as evidenced by the performance of EMVP, SelaVPR (global) and SALAD. Finally, EMVP-L obtains the best performance by leveraging the probing method tailored for the VPR task. Taking the typical MSLS Validation dataset as an example, EMVP-L even outperforms the full fine-tuning method SALAD by 1.7% at Recall@1. Additional visualization in Figure 4 shows that the VPR model fine-tuned by EMVP-B successfully finds the closest match in challenging scenarios, including occlusion, illumination change, perspective change, and seasonal variation. For more comparisons with SOTA methods, please refer to Appendix A.1.

| Query | Ground Truth | Top 1 | Top 2 | Top 3 |

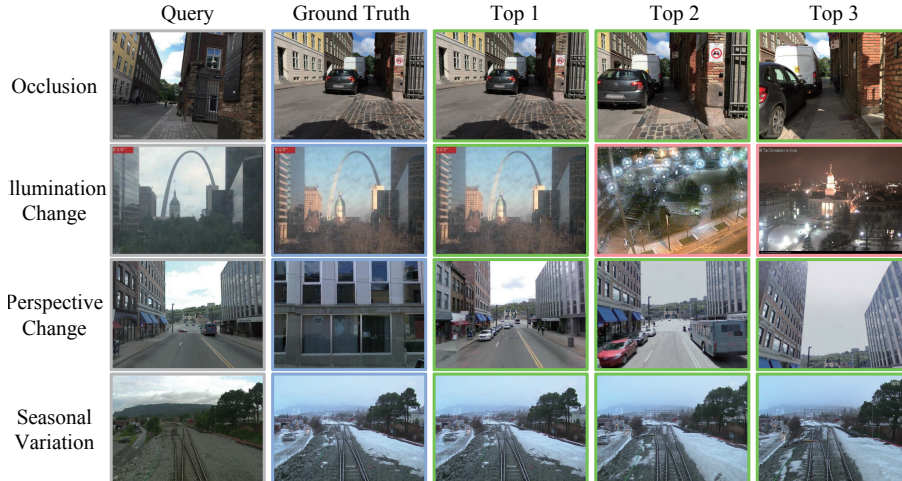

Figure 4: Query (gray) and top 3 retrieved frames (green: successful, red: failed). Moreover, one of the true (blue) matches is displayed for comparison.

## 4.3 Ablation Studies

**The Impact of Different Probings.** We compared probing methods based on first-order and second-order features. Among the first-order methods, the popular LP in classification and GeM pooling in VPR tasks are explored. For second-order feature methods, we dive into the pioneering second-order method MP in classification and the proposed CFP. Additionally, the *baseline* can be seen as a form of bilinear pooling with the centroids removed, and it is implemented by removing the optimal transport operation in the SALAD code base.

As shown in Table 2, the test results of LP revealed a significant decrease in performance compared to aggregation based on CFP. One possible reason for this could be the common trick used in LP-based methods, where average pooling is applied to local descriptors to obtain a fixed-size global one. This first-order statistical features extraction may lead to information loss. GeM pooling generalizes average and max pooling, as a result, its accuracy is still fundamentally limited by first-order features.

Despite that MP has achieved excellent accuracy metrics in classification, its performance in VPR tasks is inferior to CFP, as shown in Table 2. This is mainly due to CFP implicitly leveraging the priors provided by semantic centroids. By comparing results of baseline and NetVLAD (ID 2), we find that directly removing centroids using bilinear pooling leads to a performance drop. This is why SALAD employs optimal transport to improve the performance. In contrast, this paper introduce CN and $\text{DPN}_C$ to theoretically refine the simplification of NetVLAD, achieving improved performance.

It is noteworthy that advanced methods such as TransVPR and MixVPR, employing heavy Transformer and MLP-Mixer aggregation architectures, have demonstrated excellent performance in VPR tasks. However, these methods do not quite align with the current research paradigm of fine-tuning VFMs based on shallow trainable MLP architectures.

**The Impact of the Reinterpretation with Constant Normalization.** By comparing the experimental results of ID 2 and ID 10 in Table 2, we can verify that our reinterpretation for NetVLAD has led to improved performance. This is mainly attributed to the elimination of the explicit learning of cluster centers, which has reduced parameter number and mitigated the impact of imprecise initialization. It is worth noting that increasing the feature dimension of NetVLAD can significantly enhance performance. However, it is essential to consider the potential cost when dealing with the storage of images with sizable global descriptors, particularly in the context of VPR applications.

Comparisons between ID 7 and ID 8, ID 7 and ID 9 in Table 2 demonstrate that the CN makes this reinterpretation operation empirically more robust, and this can be validated by achieving improved performance. Through comparing ID 8 and ID 9 in Table 2, we can observe that the improvement brought by CN is dependent on its specific implementation. Further exploration of this aspect will be undertaken in our subsequent research, extending beyond the scope of this paper.

Table 2: Comparing different backbones and probings. LP, MP, CFP, CN, and $DPN_C$ indicate linear probing, moment probing, centroid-free probing, constant normalization, and dynamic power normalization in probing, respectively. For fairness, results produced by ViT-based models are obtained by fully fine-tuning the last 4 blocks. *Baseline* refers to the simplified NetVLAD adapted by SALAD. The best and the second best results are **bolded** and underlined, respectively.

| ID | Method | Fea. Dim | Backbone | MSLS Val | | | NordLand | | | Pitts250k-test | | | SPED | | |
|---|---|---|---|---|---|---|---|---|---|---|---|---|---|---|---|
| | | | | R@1 | R@5 | R@10 | R@1 | R@5 | R@10 | R@1 | R@5 | R@10 | R@1 | R@5 | R@10 |
| 1 | NetVLAD | 32768 | ResNet50 | 82.6 | 89.6 | 92.0 | 32.6 | 47.1 | 53.3 | 90.5 | 96.2 | 97.4 | 78.7 | 88.3 | 91.4 |
| 2 | NetVLAD | 8192 | ViT-B | 90.1 | 95.4 | 96.8 | 70.1 | 86.5 | 90.2 | 95.4 | 98.4 | 99.1 | 90.6 | 95.4 | 96.7 |
| 3 | NetVLAD | 24576 | ViT-B | 92.4 | 95.9 | **96.9** | 71.8 | 86.5 | 90.1 | 95.6 | 98.7 | 99.3 | 90.8 | 95.7 | 96.7 |
| 4 | LP | 768+256 | ViT-B | 85.3 | 93.5 | 95.4 | 38.1 | 55.3 | 61.8 | 91.3 | 96.9 | 98.1 | 83.0 | 92.3 | 94.0 |
| 5 | MP | 2048 | ViT-B | 87.3 | 94.5 | 96.4 | 42.6 | 62.6 | 70.0 | 92.5 | 97.3 | 98.5 | 85.2 | 92.6 | 94.6 |
| 6 | GeM | 4096 | ViT-B | 85.4 | 93.9 | 95.0 | 35.4 | 52.5 | 59.6 | 89.5 | 96.5 | 98.0 | 83.0 | 92.1 | 93.9 |
| 7 | Baseline | 8192+256 | ViT-B | 90.3 | 95.7 | 96.1 | 56.5 | 73.0 | 78.6 | 94.4 | 98.4 | 99.1 | 88.0 | 94.7 | 95.6 |
| 8 | + CN=Softmax | - | - | 91.3 | 95.7 | 96.4 | 68.0 | 82.0 | 86.2 | 94.9 | 98.3 | 99.0 | 89.3 | 94.9 | 96.4 |
| 9 | + CN= $\ell_2$ norm. | - | - | 90.8 | 95.9 | 96.6 | 66.4 | 80.9 | 84.5 | 94.5 | 98.1 | 99.0 | 89.0 | 93.9 | 95.7 |
| 10 | + $DPN_C$ (*i.e.*, CFP) | - | - | **92.6** | **96.2** | 96.8 | **74.6** | **87.6** | **91.3** | 95.2 | 98.7 | 99.3 | **92.1** | **95.9** | **97.2** |

Table 3: Comparing different fine-tuning methods. $DPN_C$ and $DPN_R$ indicate DPN in CFP and recalibration, respectively. Results of both parallel and sequential versions of $DPN_R$ are reported. For fairness, only the last 4 blocks can be fine-tuned, and all methods employ the same backbone, *i.e.*, ViT-B. The best and the second best results are **bolded** and underlined, respectively.

| Method | Fine-tuning Type | Params. (M) | MSLS Val | | | NordLand | | | Pitts250k-test | | | SPED | | |
|---|---|---|---|---|---|---|---|---|---|---|---|---|---|---|
| | | | R@1 | R@5 | R@10 | R@1 | R@5 | R@10 | R@1 | R@5 | R@10 | R@1 | R@5 | R@10 |
| AnyLoc | Zero-shot | - | 68.7 | 78.2 | 81.8 | 16.1 | 25.4 | 30.4 | 87.2 | 94.4 | 96.5 | 85.3 | 94.4 | 95.4 |
| SALAD | Full | 27.1 | 92.2 | 96.4 | 97.0 | 76.0 | **89.2** | 92.0 | 95.1 | 98.5 | 99.1 | **92.1** | 96.2 | 96.5 |
| CFP | +PSRP | 0.14 | 92.7 | 96.6 | 96.9 | 73.0 | 86.5 | 89.5 | 95.3 | 98.6 | 99.2 | 91.3 | 95.9 | 96.9 |
| | + $DPN_R$(para.) | 0.05 | 92.4 | 96.5 | 96.8 | 71.8 | 85.2 | 88.9 | 95.4 | 98.5 | 99.1 | 91.3 | 96.2 | 96.7 |
| (*i.e.*, EMVP-B) | + $DPN_R$(seq.) | 0.05 | **93.2** | **96.9** | **97.2** | **76.4** | 88.8 | **92.1** | **95.7** | **98.9** | **99.3** | 91.8 | **96.5** | **97.4** |

**The Impact of the Foundation Model.** The comparative results between ID 1 and ID 3 in Table 2 indicate that, when the size of the global descriptor is within the same order of magnitude, the VFM (*i.e.*, DINOv2) is more suitable as the backbone network for VPR tasks compared to traditional CNN models. Even in the scenario of zero-shot inference, the VPR model based on DINOv2 demonstrates strong generalization and robustness (refer to AnyLoc in Table 3). However, it is important to note that the performance of zero-shot inference still exhibits a substantial performance gap in comparison to methods that are based on fine-tuning. This motivates us to explore a more effective fine-tuning pipeline tailored for VPR tasks. Additionally, the results in Table 10 of Appendix A.2 indicate that as the scale of ViTs increases, the performance of VPR models fine-tuned by EMVP can be improved.

**Comparison of Fine-tuning Methods.** In Table 3, different fine-tuning approaches are compared. By comparing SALAD and AnyLoc, we can conclude that current VFMs (*i.e.*, DINOv2) lack sufficient zero-shot reasoning capabilities for diverse data in the VPR domain. SALAD achieves high performance by fully fine-tuning on DINOv2, but VPR models are typically deployed on mobile robots, and this full-parameter update approach imposes the higher demands on communication. Therefore, we attempt to study adaptation methods more suitable for VPR tasks.

For fairness, this paper reimplements the advanced PEFT method PSRP [42], which is also aimed at cooperating with second-order features, on the VPR dataset to compare it with our proposed $DPN_R$. Furthermore, as illustrated in Figure 3, the $DPN_R$ module can be further divided into parallel and sequential versions for comparative research. By further comparing the sequential and parallel $DPN_R$ modules in Table 3, we find that the sequential version performs better. This is primarily because the sequential method recalibrates the backbone features more thoroughly, and it does not significantly increase training difficulty since a few additional parameters are introduced. The sequential configuration of the $DPN_R$ method is used by default in other experiments in this paper. Compared with methods such as SALAD and PSRP, the sequential $DPN_R$ outperforms them by achieving the best performance while saving 64.3% of trainable parameters (0.14M *vs* 0.05M). We further visualize the impact of $DPN_R$ on recalibration in Appendix A.3.

## 5 Conclusion

In this paper, we have proposed a novel fine-tuning pipeline named EMVP, which involves reinterpreting the classical aggregation (*i.e.*, NetVLAD) into a CFP stage when fine-tuning a VFM for accurate VPR. What is more innovative, both the recalibration and CFP stages employ the same

DPN module for task-specific information preservation, effectively conducting PEFT. Extensive experiments conducted on VPR datasets have demonstrated that EMVP can extract more task-specific features, resulting in enhanced accuracy and robustness in VPR performance.

**Broader Impacts.** This work enhances the safety and efficiency of mobile robots operating in GPS-denied environments. In addition, the potential negative impact is that bad weather and ambiguous scenes will cause certain interference, making it difficult to maintain a high level of accuracy in VPR.

**Limitations.** Based on the discussions and comparisons of VFMs such as CLIP, SAM, DINO, and DINOv2 in pioneering works [48; 34; 58], this paper prioritizes DINOv2 as the VFM for experimental analysis. Note that, different VFMs possess distinct capabilities. For example, CLIP can connect images with texts for better interpretability, while SAM demonstrates powerful abilities in handling visual prompts. In the future work, we aim to explore these capabilities in different VPR tasks.

## Acknowledgements

This research was partially supported by the Zhejiang Provincial Key R&D Program of China under Grant No. 2024C01166, the Fundamental Research Funds for the Central Universities under Grant No. 226-2024-00227, the National Natural Science Foundation of China under Grant No. 62303428, and the GuangZhou City's Key R&D Program of China under Grant No. 2024B01J1301.

## References

[1] Xuecheng Xu, Sha Lu, Jun Wu, Haojian Lu, Qiuguo Zhu, Yiyi Liao, Rong Xiong, and Yue Wang. Ring++: Roto-translation-invariant gram for global localization on a sparse scan map. *IEEE Trans. Robot.*, 2023.

[2] Weinan Chen, Changfei Fu, Lei Zhu, Shing-Yan Loo, and Hong Zhang. Rumination meets vslam: You do not need to build all the submaps in realtime. *IEEE Trans. Ind. Electron.*, 2023.

[3] Haoang Li, Ji Zhao, Jean-Charles Bazin, Pyojin Kim, Kyungdon Joo, Zhenjun Zhao, and Yun-Hui Liu. Hong kong world: Leveraging structural regularity for line-based slam. *IEEE Trans. Pattern Anal. Mach. Intell.*, 2023.

[4] Relja Arandjelovic, Petr Gronat, Akihiko Torii, Tomas Pajdla, and Josef Sivic. Netvlad: Cnn architecture for weakly supervised place recognition. In *IEEE Conf. Comput. Vis. Pattern Recog.*, pages 5297–5307, 2016.

[5] Stephen Hausler, Sourav Garg, Ming Xu, Michael Milford, and Tobias Fischer. Patch-netvlad: Multi-scale fusion of locally-global descriptors for place recognition. In *IEEE Conf. Comput. Vis. Pattern Recog.*, pages 14141–14152, 2021.

[6] Yixiao Ge, Haibo Wang, Feng Zhu, Rui Zhao, and Hongsheng Li. Self-supervising fine-grained region similarities for large-scale image localization. In *Eur. Conf. Comput. Vis.*, pages 369–386. Springer, 2020.

[7] Liu Liu, Hongdong Li, and Yuchao Dai. Stochastic attraction-repulsion embedding for large scale image localization. In *Int. Conf. Comput. Vis.*, pages 2570–2579, 2019.

[8] Guohao Peng, Jun Zhang, Heshan Li, and Danwei Wang. Attentional pyramid pooling of salient visual residuals for place recognition. In *Int. Conf. Comput. Vis.*, pages 885–894, 2021.

[9] Hyo Jin Kim, Enrique Dunn, and Jan-Michael Frahm. Learned contextual feature reweighting for image geo-localization. In *IEEE Conf. Comput. Vis. Pattern Recog.*, pages 2136–2145, 2017.

[10] Ruotong Wang, Yanqing Shen, Weiliang Zuo, Sanping Zhou, and Nanning Zheng. Transvpr: Transformer-based place recognition with multi-level attention aggregation. In *IEEE Conf. Comput. Vis. Pattern Recog.*, pages 13648–13657, 2022.

[11] Sijie Zhu, Linjie Yang, Chen Chen, Mubarak Shah, Xiaohui Shen, and Heng Wang. R2former: Unified retrieval and reranking transformer for place recognition. In *IEEE Conf. Comput. Vis. Pattern Recog.*, pages 19370–19380, 2023.

[12] Sergio Izquierdo and Javier Civera. Optimal transport aggregation for visual place recognition. 2024.

[13] Feng Lu, Lijun Zhang, Xiangyuan Lan, Shuting Dong, Yaowei Wang, and Chun Yuan. Towards seamless adaptation of pre-trained models for visual place recognition. In *Int. Conf. Learn. Represent.*

[14] Peihua Li, Jiangtao Xie, Qilong Wang, and Wangmeng Zuo. Is second-order information helpful for large-scale visual recognition? In *Int. Conf. Comput. Vis.*, pages 2070–2078, 2017.

[15] Filip Radenović, Giorgos Tolias, and Ondřej Chum. Fine-tuning cnn image retrieval with no human annotation. *IEEE Trans. Pattern Anal. Mach. Intell.*, 41(7):1655–1668, 2018.

[16] Amar Ali-Bey, Brahim Chaib-Draa, and Philippe Giguere. Mixvpr: Feature mixing for visual place recognition. In *IEEE Winter Conf. Appl. Comput. Vis.*, pages 2998–3007, 2023.

[17] Gabriele Berton, Carlo Masone, and Barbara Caputo. Rethinking visual geo-localization for large-scale applications. In *IEEE Conf. Comput. Vis. Pattern Recog.*, pages 4878–4888, 2022.

[18] María Leyva-Vallina, Nicola Strisciuglio, and Nicolai Petkov. Data-efficient large scale place recognition with graded similarity supervision. In *IEEE Conf. Comput. Vis. Pattern Recog.*, pages 23487–23496, 2023.

[19] Gabriele Berton, Gabriele Trivigno, Barbara Caputo, and Carlo Masone. Eigenplaces: Training viewpoint robust models for visual place recognition. In *Int. Conf. Comput. Vis.*, pages 11080–11090, 2023.

[20] Gabriella Csurka, Christopher Dance, Lixin Fan, Jutta Willamowski, and Cédric Bray. Visual categorization with bags of keypoints. In *Eur. Conf. Comput. Vis. Worksh.*, volume 1, pages 1–2. Prague, 2004.

[21] James Philbin, Ondrej Chum, Michael Isard, Josef Sivic, and Andrew Zisserman. Object retrieval with large vocabularies and fast spatial matching. In *IEEE Conf. Comput. Vis. Pattern Recog.*, pages 1–8. IEEE, 2007.

[22] Sivic and Zisserman. Video google: A text retrieval approach to object matching in videos. In *Int. Conf. Comput. Vis.*, pages 1470–1477. IEEE, 2003.

[23] Yunchao Gong, Liwei Wang, Ruiqi Guo, and Svetlana Lazebnik. Multi-scale orderless pooling of deep convolutional activation features. In *Eur. Conf. Comput. Vis.*, pages 392–407. Springer, 2014.

[24] Hervé Jégou, Matthijs Douze, Cordelia Schmid, and Patrick Pérez. Aggregating local descriptors into a compact image representation. In *2010 IEEE computer society conference on computer vision and pattern recognition*, pages 3304–3311. IEEE, 2010.

[25] Relja Arandjelovic and Andrew Zisserman. All about vlad. In *IEEE Conf. Comput. Vis. Pattern Recog.*, pages 1578–1585, 2013.

[26] Tommi Jaakkola and David Haussler. Exploiting generative models in discriminative classifiers. *Adv. Neural Inform. Process. Syst.*, 11, 1998.

[27] Florent Perronnin, Yan Liu, Jorge Sánchez, and Hervé Poirier. Large-scale image retrieval with compressed fisher vectors. In *2010 IEEE computer society conference on computer vision and pattern recognition*, pages 3384–3391. IEEE, 2010.

[28] Hervé Jégou, Florent Perronnin, Matthijs Douze, Jorge Sánchez, Patrick Pérez, and Cordelia Schmid. Aggregating local image descriptors into compact codes. *IEEE Trans. Pattern Anal. Mach. Intell.*, 34(9):1704–1716, 2011.

[29] David G Lowe. Distinctive image features from scale-invariant keypoints. *Int. J. Comput. Vis.*, 60:91–110, 2004.

[30] Herbert Bay, Andreas Ess, Tinne Tuytelaars, and Luc Van Gool. Speeded-up robust features (surf). *Comput Vis Image Underst*, 110(3):346–359, 2008.

[31] Giorgos Tolias, Ronan Sicre, and Hervé Jégou. Particular object retrieval with integral max-pooling of cnn activations. 2016.

[32] Qibo Qiu, Wenxiao Wang, Haochao Ying, Dingkun Liang, Haiming Gao, and Xiaofei He. Selfloc: Selective feature fusion for large-scale point cloud-based place recognition. *Knowl-Based Syst.*, 295:111794, 2024.

[33] Amar Ali-bey, Brahim Chaib-draa, and Philippe Giguère. Gsv-cities: Toward appropriate supervised visual place recognition. *Neurocomputing*, 513:194–203, 2022.

[34] Nikhil Keetha, Avneesh Mishra, Jay Karhade, Krishna Murthy Jatavallabhula, Sebastian Scherer, Madhava Krishna, and Sourav Garg. Anyloc: Towards universal visual place recognition. *arXiv preprint arXiv:2308.00688*, 2023.

[35] Tom Brown, Benjamin Mann, Nick Ryder, Melanie Subbiah, Jared D Kaplan, Prafulla Dhariwal, Arvind Neelakantan, Pranav Shyam, Girish Sastry, Amanda Askell, et al. Language models are few-shot learners. *Advances in neural information processing systems*, 33:1877–1901, 2020.

[36] Alexander Kirillov, Eric Mintun, Nikhila Ravi, Hanzi Mao, Chloe Rolland, Laura Gustafson, Tete Xiao, Spencer Whitehead, Alexander C Berg, Wan-Yen Lo, et al. Segment anything. *arXiv preprint arXiv:2304.02643*, 2023.

[37] Maxime Oquab, Timothée Darcet, Théo Moutakanni, Huy Vo, Marc Szafraniec, Vasil Khalidov, Pierre Fernandez, Daniel Haziza, Francisco Massa, Alaaeldin El-Nouby, et al. Dinov2: Learning robust visual features without supervision. *arXiv preprint arXiv:2304.07193*, 2023.

[38] Alec Radford, Jong Wook Kim, Chris Hallacy, Aditya Ramesh, Gabriel Goh, Sandhini Agarwal, Girish Sastry, Amanda Askell, Pamela Mishkin, Jack Clark, et al. Learning transferable visual models from natural language supervision. pages 8748–8763. PMLR, 2021.

[39] Neil Houlsby, Andrei Giurgiu, Stanislaw Jastrzebski, Bruna Morrone, Quentin De Laroussilhe, Andrea Gesmundo, Mona Attariyan, and Sylvain Gelly. Parameter-efficient transfer learning for nlp. pages 2790–2799. PMLR, 2019.

[40] Menglin Jia, Luming Tang, Bor-Chun Chen, Claire Cardie, Serge Belongie, Bharath Hariharan, and Ser-Nam Lim. Visual prompt tuning. In *Eur. Conf. Comput. Vis.*, pages 709–727. Springer, 2022.

[41] Edward J Hu, Phillip Wallis, Zeyuan Allen-Zhu, Yuanzhi Li, Shean Wang, Lu Wang, Weizhu Chen, et al. Lora: Low-rank adaptation of large language models. In *Int. Conf. Learn. Represent.*, 2021.

[42] Mingze Gao, Qilong Wang, Zhenyi Lin, Pengfei Zhu, Qinghua Hu, and Jingbo Zhou. Tuning pre-trained model via moment probing. In *Int. Conf. Comput. Vis.*, pages 11803–11813, 2023.

[43] Tsung-Yu Lin, Aruni RoyChowdhury, and Subhransu Maji. Bilinear cnn models for fine-grained visual recognition. In *Int. Conf. Comput. Vis.*, pages 1449–1457, 2015.

[44] Tsung-Yu Lin and Subhransu Maji. Improved bilinear pooling with cnns. In *Brit. Mach. Vis. Conf.*, 2017.

[45] Qilong Wang, Mingze Gao, Zhaolin Zhang, Jiangtao Xie, Peihua Li, and Qinghua Hu. Dropcov: a simple yet effective method for improving deep architectures. *Adv. Neural Inform. Process. Syst.*, 35:33576–33588, 2022.

[46] Mathilde Caron, Hugo Touvron, Ishan Misra, Hervé Jégou, Julien Mairal, Piotr Bojanowski, and Armand Joulin. Emerging properties in self-supervised vision transformers. In *Int. Conf. Comput. Vis.*, pages 9650–9660, 2021.

[47] Kaiming He, Xinlei Chen, Saining Xie, Yanghao Li, Piotr Dollár, and Ross Girshick. Masked autoencoders are scalable vision learners. In *IEEE Conf. Comput. Vis. Pattern Recog.*, pages 16000–16009, 2022.

[48] Ani Vanyan, Alvard Barseghyan, Hakob Tamazyan, Vahan Huroyan, Hrant Khachatrian, and Martin Danelljan. Analyzing local representations of self-supervised vision transformers. *arXiv preprint arXiv:2401.00463*, 2023.

[49] Frederik Warburg, Soren Hauberg, Manuel Lopez-Antequera, Pau Gargallo, Yubin Kuang, and Javier Civera. Mapillary street-level sequences: A dataset for lifelong place recognition. In *IEEE Conf. Comput. Vis. Pattern Recog.*

[50] Akihiko Torii, Josef Sivic, Tomas Pajdla, and Masatoshi Okutomi. Visual place recognition with repetitive structures. In *IEEE Conf. Comput. Vis. Pattern Recog.*, pages 883–890, 2013.

[51] Daniel Olid, José M Fácil, and Javier Civera. Single-view place recognition under seasonal changes. *arXiv preprint arXiv:1808.06516*, 2018.

[52] Mubariz Zaffar, Sourav Garg, Michael Milford, Julian Kooij, David Flynn, Klaus McDonald-Maier, and Shoaib Ehsan. Vpr-bench: An open-source visual place recognition evaluation framework with quantifiable viewpoint and appearance change. *Int. J. Comput. Vis.*, 129(7):2136–2174, 2021.

[53] Zetao Chen, Lingqiao Liu, Inkyu Sa, Zongyuan Ge, and Margarita Chli. Learning context flexible attention model for long-term visual place recognition. *IEEE Robot. Autom. Lett.*, 3(4):4015–4022, 2018.

[54] Jun Yu, Chaoyang Zhu, Jian Zhang, Qingming Huang, and Dacheng Tao. Spatial pyramid-enhanced netvlad with weighted triplet loss for place recognition. *IEEE Trans. Neural Networks Learn. Syst.*, 31(2):661–674, 2019.

[55] Jian Zhang, Yunyin Cao, and Qun Wu. Vector of locally and adaptively aggregated descriptors for image feature representation. *Pattern Recognition*, 116:107952, 2021.

[56] Paul-Edouard Sarlin, Daniel DeTone, Tomasz Malisiewicz, and Andrew Rabinovich. Superglue: Learning feature matching with graph neural networks. In *IEEE Conf. Comput. Vis. Pattern Recog.*, pages 4938–4947, 2020.

[57] Bingyi Cao, Andre Araujo, and Jack Sim. Unifying deep local and global features for image search. In *Eur. Conf. Comput. Vis.*, pages 726–743. Springer, 2020.

[58] Timothée Darcet, Maxime Oquab, Julien Mairal, and Piotr Bojanowski. Vision transformers need registers. *Int. Conf. Learn. Represent.*, 2023.

# A  Technical Appendix

## A.1  Implementation Details and More Comparisons

The implementation details are reported in Table 4. To facilitate a fair comparison, we try to keep the experiment settings as consistent as possible with the comparative methods. We further provide a detailed implementation of the Dynamic Power Normalization (DPN), as shown by Algorithm 1. The VPR model is fine-tuned on the GSV-Cities dataset, which contains $0.56$ million images from $67$ thousand different places. Table 5 describes the details of various testing datasets. All experiments are conducted on a NVIDIA RTX A6000 GPU using PyTorch. Moreover, fine-tuning a VPR model based on the ViT-B takes 7 minutes per epoch, and requires 21GB GPU memory. Table 6 provides a more comprehensive evaluation of single- and two-stage methods.

Table 4: Experiment setting for fine-tuning.

| Config | Value |
|---|---|
| Precision | 16-mixed |
| Optimizer | AdamW |
| Learning rate | 1e-3 |
| Weight decay | 1e-9 |
| Batch size | 480 |
| Places | 120 |
| Images per place | 4 |
| Image size | $224 \times 224$ |
| Number of patches (L) | $16 \times 16$ |
| Patch size | $14 \times 14$ |
| Output size of $\mathcal{F}_C$ (D) | 128 |
| Output size of $\mathcal{F}_P$ (K) | 64 |
| Number of epochs | 30 |

Table 5: Datasets description.

| Dataset | Database | Queries | Viewpoint | Season |
|---|---|---|---|---|
| MSLS Val | 18871 | 740 | ✓ | ✓ |
| Pitts250k-test | 83952 | 8280 | ✓ | ✗ |
| Nordland | 27592 | 2760 | ✗ | ✓ |
| SPED | 607 | 607 | ✗ | ✓ |

---

**Algorithm 1** PyTorch Pseudo Code for DPN.

---

```
1  class DPN(nn.Module):
2    def __init__(self, D, d, eps):
3      super(DPN, self).__init__()
4      self.avg_pool = nn.AdaptiveAvgPool1d(1)
5      # D: Dimension of each local descriptor.
6      self.projection = nn.Sequential(
7        nn.Linear(D,d),
8        nn.Dropout(0.1),
9        nn.ReLU(),
10       nn.Linear(d, 1)
11       )
12     self.activation = nn.Sigmoid()
13     self.eps = eps
14     def forward(self, G):
15       # G: Fine-grained features to be normalized.
16       # L: Number of local descriptors in G.
17       # D: Dimension of each local descriptor.
18       _, L, D = G.shape
19       avg_G = self.avg_pool(G.transpose(-1,-2)).squeeze(-1)
20       proj = self.projection(avg_G)
21       p = self.activation(proj).unsqueeze(-1)
22       sign = torch.sign(G)
23       pow_G = torch.pow(torch.abs(G) + self.eps, p.expand(-1,L,D))
24       return sign * pow_G + G
```

---

Table 6: More comparisons with state-of-the-art methods. The results marked with $\gamma$ are reproduced based on the official code.

| Method | MSLS Val | | | NordLand★ [52] | | | Pitts250k-test | | | SPED | | | Pitts30k-test | | | NordLand★★ [51] | | |
|---|---|---|---|---|---|---|---|---|---|---|---|---|---|---|---|---|---|---|
| | R@1 | R@5 | R@10 | R@1 | R@5 | R@10 | R@1 | R@5 | R@10 | R@1 | R@5 | R@10 | R@1 | R@5 | R@10 | R@1 | R@5 | R@10 |
| TransVPR w/o re-ranking [10] | 70.8 | 85.1 | 89.6 | $10.3^\gamma$ | $20.9^\gamma$ | $27.2^\gamma$ | $70.1^\gamma$ | $85.2^\gamma$ | $89.1^\gamma$ | $55.7^\gamma$ | $69.7^\gamma$ | $76.1^\gamma$ | 73.8 | 88.1 | 91.9 | 15.9 | 38.6 | 49.4 |
| SPE-VLAD♭ [54] | 78.2 | 86.8 | 88.8 | 25.5 | 40.1 | 46.1 | 89.9 | 96.1 | 97.3 | 73.1 | 85.5 | 88.7 | - | - | - | - | - | - |
| Gated NetVLAD♭ [55] | 82.0 | 88.9 | 91.4 | 34.4 | 50.4 | 57.7 | 89.7 | 95.9 | 97.1 | 75.6 | 87.1 | 90.8 | - | - | - | - | - | - |
| NetVLAD♭ [4] | 82.6 | 89.6 | 92.0 | 32.6 | 47.1 | 53.3 | 90.5 | 96.2 | 97.4 | 78.7 | 88.3 | 91.4 | $90.0^\gamma$ | $95.1^\gamma$ | $96.4^\gamma$ | $69.4^\gamma$ | $85.9^\gamma$ | $91.1^\gamma$ |
| Conv-AP♭ [33] | 83.4 | 90.5 | 92.3 | 38.2 | 54.8 | 61.2 | 92.4 | 97.4 | 98.4 | 80.1 | 90.3 | 93.6 | $90.6^\gamma$ | $95.1^\gamma$ | $96.2^\gamma$ | $67.7^\gamma$ | $85.1^\gamma$ | $90.5^\gamma$ |
| CosPlace♭ [17] | 83.0 | 89.9 | 91.8 | 34.4 | 49.9 | 56.5 | 91.5 | 96.9 | 97.9 | 75.3 | 85.9 | 88.6 | $90.4^\gamma$ | $95.3^\gamma$ | $96.3^\gamma$ | $70.5^\gamma$ | $87.3^\gamma$ | $93.2^\gamma$ |
| MixVPR♭ [16] | 88.0 | 92.7 | 94.6 | 58.4 | 74.6 | 80.0 | 94.6 | 98.3 | 99.0 | 85.2 | 92.1 | 94.6 | $91.6^\gamma$ | $95.5^\gamma$ | $96.4^\gamma$ | $80.4^\gamma$ | $92.4^\gamma$ | $95.9^\gamma$ |
| EigenPlaces [19] | 89.3 | 93.7 | 95.0 | 54.4 | 68.8 | 74.1 | 94.1 | 98.0 | 98.7 | 69.9 | 82.9 | 87.6 | $92.5^\gamma$ | $96.7^\gamma$ | $97.6^\gamma$ | $65.9^\gamma$ | $81.9^\gamma$ | $87.9^\gamma$ |
| SelaVPR (gobal) [13] | 87.7 | 95.8 | 96.6 | $43.8^\gamma$ | $63.1^\gamma$ | $71.1^\gamma$ | $92.6^\gamma$ | $98.0^\gamma$ | $98.8^\gamma$ | $83.5^\gamma$ | $92.6^\gamma$ | $94.6^\gamma$ | 90.2 | 96.1 | 97.1 | 72.3 | 89.4 | 94.4 |
| SALAD♭ [12] | 92.2 | 96.4 | 97.0 | 76.0 | 89.2 | 92.0 | 95.1 | 98.5 | 99.1 | 92.1 | 96.2 | 96.5 | $92.4^\gamma$ | $96.3^\gamma$ | $97.4^\gamma$ | $89.4^\gamma$ | $97.2^\gamma$ | $99.0^\gamma$ |
| **EMVP-L♭ (Ours)** | **93.9** | **97.3** | **97.6** | **78.4** | **89.7** | **92.4** | **96.5** | **99.1** | **99.5** | **94.6** | **97.5** | **98.4** | **94.0** | **97.5** | **98.2** | 88.7 | **97.3** | **99.3** |
| SP-SuperGlue♯ [56] | 78.1 | 81.9 | 84.3 | $29.0^\gamma$ | $34.8^\gamma$ | $35.8^\gamma$ | $91.4^\gamma$ | $96.5^\gamma$ | $97.4^\gamma$ | $81.4^\gamma$ | $90.0^\gamma$ | $91.8^\gamma$ | 87.2 | 94.8 | 96.4 | 25.8 | 35.4 | 38.2 |
| Patch NetVLAD♯ [5] | 79.5 | 86.2 | 87.7 | $30.8^\gamma$ | $34.5^\gamma$ | $35.4^\gamma$ | $90.9^\gamma$ | $96.1^\gamma$ | $97.2^\gamma$ | $87.6^\gamma$ | $93.6^\gamma$ | $95.1^\gamma$ | 88.7 | 94.5 | 95.9 | 51.6 | 60.1 | 62.8 |
| DELG♯ [57] | 83.2 | 90.0 | 91.1 | - | - | - | - | - | - | - | - | - | 89.9 | 95.4 | 96.7 | 51.3 | 66.8 | 69.8 |
| TransVPR♯ [10] | 86.8 | 91.2 | 92.4 | $46.9^\gamma$ | $51.6^\gamma$ | $52.5^\gamma$ | $88.8^\gamma$ | $94.2^\gamma$ | $95.2^\gamma$ | $85.3^\gamma$ | $90.8^\gamma$ | $91.3^\gamma$ | 89.0 | 94.9 | 96.2 | 58.8 | 75.0 | 78.7 |
| R2Former♯ [11] | 89.7 | 95.0 | 96.2 | $40.2^\gamma$ | $46.3^\gamma$ | $47.3^\gamma$ | $93.1^\gamma$ | $97.4^\gamma$ | $98.4^\gamma$ | $67.5^\gamma$ | $75.8^\gamma$ | $77.8^\gamma$ | 91.1 | 95.2 | 96.3 | $77.0^\gamma$ | $89.0^\gamma$ | $91.9^\gamma$ |
| SelaVPR♯ [13] | 90.8 | 96.4 | 97.2 | $63.0^\gamma$ | $77.6^\gamma$ | $81.4^\gamma$ | 95.7 | 98.8 | 99.2 | $89.8^\gamma$ | $94.7^\gamma$ | $96.1^\gamma$ | 92.8 | 96.8 | 97.7 | 85.2 | 95.5 | 98.5 |

## A.2 More Ablation Studies

**Dimensions of $\mathcal{F}_C$ and $\mathcal{F}_P$.** Table 7 and 8 indicate that the output sizes of $\mathcal{F}_C$ and $\mathcal{F}_P$ have some influence on performance, while the performance is not particularly sensitive to these two hyperparameters. When the output size of $\mathcal{F}_C$ ($D$) ranges from 128 to 512 and the output size of $\mathcal{F}_P$ ($K$) ranges from 64 to 256, the models all achieve good performance.

**Numbers of Recalibrated Blocks.** Table 9 shows that the number of blocks inserted DPN modules significantly affects the results, and the best results are achieved when the features in the last 4 blocks are recalibrated. However, the selection of recalibrated blocks is beyond the scope of this paper, and we will further explore it in future work.

**The Scale of ViT Model.** Table 10 displays the results of different ViT models at size S, B, and L. As the scale of ViT models increases, the total number of parameters grows exponentially, making training with full fine-tuning methods extremely challenging. For example, if the ViT-B is replaced with the ViT-L in the full fine-tuning method SALAD, there would be a significant decrease in evaluation performance [12]. However, thanks to the introduction of the $\text{DPN}_\text{R}$ module, the number of trainable parameters in EMVP only slightly increases with the scale of the ViT, allowing for the full utilization of the enhanced representation capability brought by the larger model size. Due to the limitation in computational resources, ViT-G is not tested.

Table 7: Impact of the output sizes of $\mathcal{F}_C$ ($K = 64$).

| D | MSLS Val | | | Pitts250k-test | | |
|---|---|---|---|---|---|---|
| | R@1 | R@5 | R@10 | R@1 | R@5 | R@10 |
| 32 | 92.0 | 96.2 | 96.8 | 95.0 | 98.4 | 99.0 |
| 64 | 92.4 | 96.4 | 96.9 | 95.5 | 98.7 | 99.2 |
| 128 | 93.2 | 96.9 | 97.2 | 95.7 | 98.9 | 99.3 |
| 256 | 92.8 | 96.8 | 97.0 | 95.5 | 98.8 | 99.4 |
| 512 | 92.7 | 96.2 | 96.9 | 95.8 | 98.8 | 99.3 |

Table 8: Impact of the output sizes of $\mathcal{F}_P$ ($D = 128$).

| K | MSLS Val | | | Pitts250k-test | | |
|---|---|---|---|---|---|---|
| | R@1 | R@5 | R@10 | R@1 | R@5 | R@10 |
| 16 | 91.5 | 96.2 | 96.5 | 94.6 | 98.3 | 99.0 |
| 32 | 91.9 | 96.2 | 96.6 | 95.2 | 98.8 | 99.3 |
| 64 | 93.2 | 96.9 | 97.2 | 95.7 | 98.9 | 99.3 |
| 128 | 93.0 | 96.5 | 97.0 | 95.8 | 98.8 | 99.4 |
| 256 | 92.8 | 96.6 | 97.0 | 95.6 | 98.8 | 99.3 |

Table 9: Impact of the number of recalibrated blocks.

| Blocks | MSLS Val | | | Pitts250k-test | | |
|---|---|---|---|---|---|---|
| | R@1 | R@5 | R@10 | R@1 | R@5 | R@10 |
| 0 | 90.9 | 95.7 | 96.4 | 94.6 | 98.2 | 99.0 |
| 2 | 92.3 | 96.4 | 96.9 | 95.1 | 98.8 | 99.2 |
| 4 | 93.2 | 96.9 | 97.2 | 95.7 | 98.9 | 99.3 |
| 6 | 92.2 | 96.4 | 96.9 | 95.2 | 98.5 | 99.0 |
| 8 | 89.2 | 94.6 | 95.8 | 94.9 | 98.3 | 99.0 |

Table 10: Comparing different ViT models. Tr. and Ttl. represent the number of trainable and total parameters (M), respectively.

| Arch. | Tr./Ttl. | MSLS Val | | | Pitts250k-test | | |
|---|---|---|---|---|---|---|---|
| | | R@1 | R@5 | R@10 | R@1 | R@5 | R@10 |
| ViT-S | 0.02/21 | 91.5 | 95.5 | 96.8 | 95.2 | 98.6 | 99.2 |
| ViT-B | 0.05/86 | 93.2 | 96.9 | 97.2 | 95.7 | 98.9 | 99.3 |
| ViT-L | 0.06/300 | 93.9 | 97.3 | 97.6 | 96.5 | 99.1 | 99.5 |

### A.3 Visualizations for $DPN_R$

The visualization results in Figure 5 indicate that after being trained by $DPN_R$, the top 20% of high-norm tokens tend to appear more in distinctive regions. Figure 6 illustrates that after being fine-tuned through the EMVP pipeline, the VPR model can accurately capture texture details shared among different images. While Figure 7 shows that under changes in perspective, high-norm tokens tend to appear in distinctive background regions, which are typically the tallest building in a place. This is primarily attributed to the $DPN_R$ module adopted by EMVP, which enhances task-specific representations while maximally preserving the feature representation capability of the VFM.

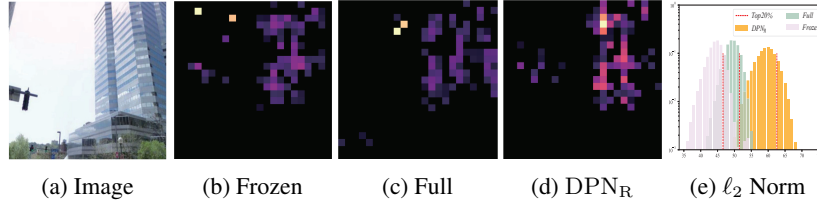

(a) Image        (b) Frozen        (c) Full        (d) $DPN_R$        (e) $\ell_2$ Norm

Figure 5: The visualization of the top 20% high-norm tokens.

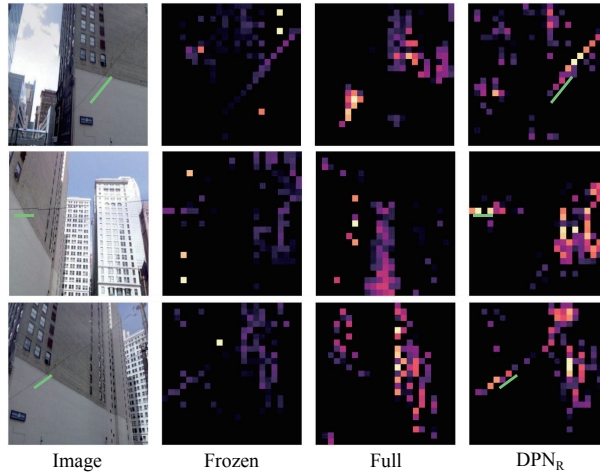

Image        Frozen        Full        $DPN_R$

Figure 6: High-norm tokens can contribute to capturing texture details.

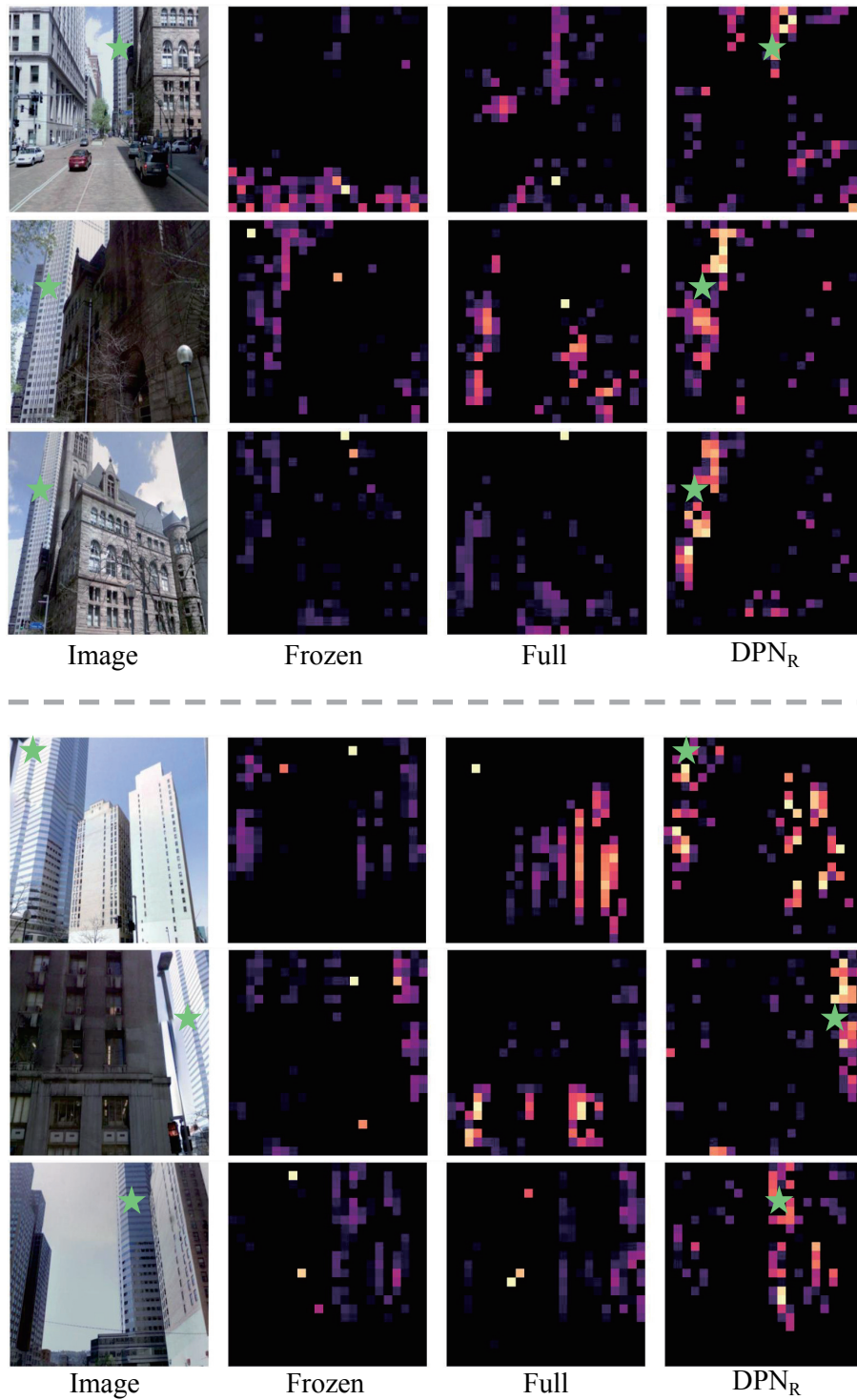

|      Image      |      Frozen      |      Full      |      DPN$_R$      |

Figure 7: Visual place recognition under changes in perspectives.

